# Selecting Diverse Features via Spectral Regularization

**Abhimanyu Das**[*]
Microsoft Research
Mountain View
abhidas@microsoft.com

**Anirban Dasgupta**
Yahoo! Labs
Sunnyvale
anirban@yahoo-inc.com

**Ravi Kumar**[*]
Google
Mountain View
ravi.k53@gmail.com

## Abstract

We study the problem of diverse feature selection in linear regression: selecting a small subset of diverse features that can predict a given objective. Diversity is useful for several reasons such as interpretability, robustness to noise, etc. We propose several spectral regularizers that capture a notion of diversity of features and show that these are all submodular set functions. These regularizers, when added to the objective function for linear regression, result in approximately submodular functions, which can then be maximized by efficient greedy and local search algorithms, with provable guarantees. We compare our algorithms to traditional greedy and $\ell_1$-regularization schemes and show that we obtain a more diverse set of features that result in the regression problem being stable under perturbations.

## 1 Introduction

Feature selection is a key component in many machine learning settings. The process involves choosing a small subset of features in order to build a model to approximate the target concept well. Feature selection offers several advantages in practice. This includes reducing the dimension of the data and hence the space requirements, enhancing the interpretability of the learned model, mitigating over-fitting, decreasing generalization error, etc.

In this paper we focus on feature selection for linear regression, which is the prediction model of choice for many practitioners. The goal is to obtain a linear model using a subset of $k$ features (where $k$ is user-specified), to minimize the prediction error or, equivalently, maximize the squared multiple correlation [16]. In general, feature selection techniques can be categorized into two approaches. In the first, features are greedily selected one by one up to the pre-specified budget $k$; the Forward or Backward greedy methods[19] fall into this type. In the second, the feature selection process is intimately coupled with the regression objective itself by adding a (usually convex) *regularizer*. For example, the *Lasso* [20] uses the $\ell_1$-norm of the coefficients as a regularizer to promote sparsity.

In this work we consider the feature selection problem of choosing the best set of features for predicting a specified target, coupled with the desire to choose as "diverse" features as possible; our goal will be to construct a regularizer that can capture diversity. Diversity among the chosen features is a useful property for many reasons. Firstly, it increases the interpretability of the chosen features, since we are assured that they not redundant and are more representative of the feature space covered by the entire dataset (see e.g. [7]). Secondly, as we show, the right notion of diversity can also make the feature selection task resistant to noise in the data. Thirdly, it is well known that correlated features can slow down the convergence of algorithms such as the stochastic gradient (e.g., [2]); by demanding diversity, one can potentially obviate this slowdown.

---

[*]This work was done while the author was at Yahoo! Labs.

Unfortunately, the traditional greedy and $\ell_1$-relaxation approaches to feature-selection do not explictly address feature diversity[1]. In this paper, we address this problem of diverse feature selection using an approach that falls between that of greedy methods and convex-regularization methods. In particular, we construct regularizers that capture a notion of diversity — unlike regularizers such as Lasso, our regularizers are set functions as opposed to functions of the regression coefficient vector. Our objective function are thus a combination of the linear regression objective and the regularizer. We then design provable approximation algorithms for such objectives using a combination of greedy and local search techniques. While there is no unique way to define feature diversity, we take a spectral approach. By defining diversity to be a carefully chosen function of the spectrum of the chosen features, we tap into notions of submodularity and consequently into the rich literature for maximizing submodular functions [5, 9, 14].

Our contributions are as follows: (i) We formulate an optimization problem for diverse feature selection and construct a family of submodular spectral regularizers that capture diversity notions. (ii) We use a novel approach of combining the diversity regularizers with the optimization objective to obtain (approximately) submodular maximization problems, and optimize them using greedy and local search algorithms with provable guarantees. (iii) We validate the performance of our algorithms using experiments on real and synthetic data sets.

## 2 Related work

Feature selection and the closely related problems of sparse approximation/recovery have been extensively studied using two broad classes of methods: greedy [5, 19, 21, 11, 24] and convex relaxation [20, 25, 3, 22, 8]. None of these methods, however, takes feature diversity into the account during selection. The (greedy) methods in our paper are inspired by those of Das and Kempe [5], who provide prediction error bounds using a notion of approximate submodularity. However, they do not incorporate any notion of feature diversity; they also require monotonicity, which does not hold for several regularizers we construct. A related convex relaxation based approach is that of Grave et al. [12], who address the unstable behavior of Lasso in the presence of correlated features and propose adding a trace norm regularizer to the error objective. The focus is to select groups of correlated variables together instead of selecting only one variable from each group. Our goal is different: select variables that are relatively uncorrelated with each other.

Previous work on diverse feature selection includes greedy heuristics for trading-off information-theoretic feature relevance and feature redundancy criteria when selecting features [7, 23]. However the heuristics presented do not carry any theoretical guarantees.

There has been some work on selecting a diverse set of features to maximize the mutual information or the entropy of a set of variables [13, 17]. But, the problem definition in these works does not specify a target prediction vector or variable; the goal instead is to select diverse features regardless of whether the features are relevant for predicting a particular target variable. On the other hand, our work requires us to simultaneously optimize for both feature selection and diversity objectives.

If we consider orthogonality as a loose proxy for diversity, methods such Principal Component Analysis and Singular Value Decomposition [15] become relevant. However, these methods do not return elements from the original set of features and instead output linear combinations of the feature vectors; this might not be desirable for many applications.

## 3 Preliminaries

For any symmetric positive semidefinite $n \times n$ matrix $A$, we denote its eigenvalues by $\lambda_{\min}(A) = \lambda_1(A) \leq \cdots \leq \lambda_n(A) = \lambda_{\max}(A)$. We use $\det(A) = \Pi_{i=1}^{n} \lambda_i(A)$ to denote the determinant of $A$. Recall the vector and matrix two-norms: $\|x\|_2 = \sqrt{\sum_i |x_i|^2}$ and $\|A\|_2 = \lambda_{\max}(A)$.

Let $X = \{X_1, \ldots, X_n\}$ be the set of feature vectors (or random variables) where each $X_i \in \mathbb{R}^m$ and let $Z \in \mathbb{R}^m$ be the target vector. By appropriate normalization, we can assume $\|X_i\|_2 = 1 = \|Z\|_2$. We wish to predict $Z$ using linear regression on a small subset of $X$. The matrix of inner products (or

covariances) between the $X_i$ and $X_j$ is denoted by $C$, with entries $C_{i,j} = \mathrm{Cov}(X_i, X_j)$. Similarly, we use $b$ to denote the inner products between $Z$ and the $X_i$'s, with $b_i = \mathrm{Cov}(Z, X_i)$.

For a $n$-dimensional Gaussian random vector $v$ with covariance matrix $C$, we use $\mathrm{H}(v) = \frac{1}{2}\log((2\pi e)^n \det(C))$ to denote the differential entropy of $v$.

For a set $S \subseteq X$, if $Z'(S)$ is the optimal linear predictor of $Z$ using the vectors in $S$, then the *squared multiple correlation* [6, 16] is defined as $R_Z^2(S) = 1 - \|(Z - Z'(S))\|_2^2$. This is a widely used goodness-of-fit measure; it captures the length of the projection of $Z$ on the subspace spanned by the vectors in $S$.

**Definition 1 (Diverse feature selection)** *Given $k > 0$, find a set $S \subseteq X$ satisfying*

$$\operatorname*{argmax}_{S:|S|\leq k} g(S) \triangleq R_Z^2(S) + \nu f(S), \tag{1}$$

*where $\nu > 0$ is the regularization constant and $f(S)$ is some "diversity-promoting" regularizer.*

Note that diversity is not a uniquely-defined notion, however, we call a regularizer $f$ to be *diversity-promoting* if the following two conditions are satisfied: for a fixed $k$, $f(S)$ is maximized when $S$ is an orthogonal set of vectors and is minimized when $S$ has the lowest rank, where $|S| \leq k$.

For convenience, we do not distinguish between the index set $S$ and the variables $\{X_i \mid i \in S\}$. We use $C_S$ to denote the submatrix of $C$ with row and column set $S$, and $b_S$ to denote the vector with only entries $b_i$ for $i \in S$. Given the subset $S$ of vectors used for prediction, the optimal regression coefficients $\alpha_i$ are $(\alpha_i)_{i \in S} = C_S^{-1} b_S$ (e.g., [16]) and hence $R_Z^2(S) = b_S^T C_S^{-1} b_S$. [2]

Many of our results are phrased in terms of eigenvalues of the inner product matrix $C$ and its submatrices. Since such matrices are positive semidefinite, their eigenvalues are real, non-negative [16].

**Submodularity ratio.** Das and Kempe [5] introduced the notion of *submodularity ratio* for a general set function to capture how close is the function to being submodular.

**Definition 2 (Submodularity ratio)** *Let $f$ be a non-negative set function. The submodularity ratio of $f$ with respect to a set $U$ and a parameter $k \geq 1$ is*
$$\gamma_{U,k}(f) = \min_{L \subseteq U, S:|S|\leq k, S\cap L=\emptyset} \frac{\sum_{x\in S} f(L \cup \{x\}) - f(L)}{f(L \cup S) - f(L)}.$$

Thus, it captures how much $f$ can increase by adding any subset $S$ of size $k$ to $L$, compared to the combined benefits of adding its individual elements to $L$. In particular, [5] defines the submodularity ratio for the $R^2$ function and relates it to the smallest eigenvalue of the covariance matrix of the data. They also show that, in practice, the submodularity ratio for $R^2$ is often quite close to 1, and hence a greedy algorithm is a good approximation to maximizing $R^2$ subject to a cardinality constraint.

**Theorem 3 (from [4])** *Let $f$ be a non-negative, monotone set function and let* OPT *be the maximum value of $f$ value obtained by any set of size $k$. Then, the set $\tilde{S}$ selected by the Greedy Algorithm has the following approximation guarantee: $f(\tilde{S}) \geq (1 - e^{-\gamma_{\tilde{S},k}(f)}) \cdot$ OPT.*

## 3.1 Robustness to perturbations

As mentioned earlier, in addition to providing better interpretability, another benefit of diverse feature selection is robustness to feature and label perturbations. Given a selected subset $S$, we now obtain a connection between the robustness of the estimated regression coefficients and the spectrum of $C_S$, in the presence of noise. Suppose $S$, a subset of size $k$, is used to predict the target vector $Z \in \mathbb{R}^n$. Let $A \in \mathbb{R}^{n \times k}$ be the vectors from $X$ corresponding to $S$. Then $C_S = A^T A$ and the optimal regression coefficients are $\alpha = C_S^{-1} A^T Z$.

Now suppose the target vector is perturbed with an i.i.d. Gaussian noise, i.e., $Z' = Z + \eta$, where $\eta \sim N(0, \sigma^2 I_n)$ is a random vector corresponding to measurement errors. Let the corresponding

regression coefficient vector be $\alpha' = C_S^{-1} A^T Z'$. We show the following perturbation result relating the differential entropy of the perturbation error in the regression coefficients to the spectrum of $C_S$.

**Lemma 4** $\mathrm{H}(\alpha' - \alpha) = k \log(2\sigma^2 \pi e) - \sum_{i=1}^{k} \log(\lambda_i(C_S))$.

**Proof.** Let $\delta = \alpha' - \alpha = C_S^{-1} A^T \eta$. Since $\eta \sim N(0, \sigma^2 I_{n \times n})$, we have that $\delta \sim N(0, C_S^{-1} A^T \cdot \sigma^2 I_{n \times n} \cdot (C_S^{-1} A^T)^T)$. Or, $\delta \sim N(0, \sigma^2 C_S^{-1})$. Thus, $\mathrm{H}(\delta) = \log((2\sigma^2 \pi e)^k \det(C_S^{-1})) = k \log(2\sigma^2 \pi e) - \sum_{i=1}^{k} \log(\lambda_i(C_S))$. ∎

Thus the perturbation error entropy is minimized by maximizing $\sum_{i=1}^{k} \log(\lambda_i(C_S))$, which motivates the smoothed differential-entropy regularizer used in Section 5.1.

We can also show (supplementary material) that the two-norm of the perturbation error in the regression coefficients is also related to the spectrum of $C_S$: the expected noise in the regression coefficients depends on the sum of the eigenvalues of $C_S^{-1}$. This suggests the use of $-\sum_i \frac{1}{\lambda_i(C_S)}$ as a diversity-promoting regularizer in Definition 1. Unfortunately, this regularization function is not submodular and is thus hard to use directly. However, as seen in Sections 5.2 and 5.3, there are other related spectral functions that are indeed submodular and can thus be used as efficient regularizers.

## 4 Algorithms

In this section we present a greedy and local-search based (GLS) approximation algorithm for solving (1) when $f(S)$ is a non-negative (but not necessarily monotone) submodular function (w.l.o.g., $\nu = 1$). In order to give an approximation algorithm for $\mathrm{argmax}_{S:|S| \leq k} g(S)$, we need to follow a sequence of steps. First we show a technical result (Theorem 5) that says that though the approximation guarantees of [5] do not carry over to the non-monotone case, we can still prove a weaker result that relates the solution obtained by a greedy algorithm with any feasible solution, as long as $g(S)$ is approximately submodular and non-negative (which holds if $f(S)$ is a non-negative submodular function). Next, we modify a local-search based algorithm for unconstrained submodular maximization to give an approximation of $\mathrm{argmax}_S g(S)$ (Theorem 7). We put these together using the framework of [9] to show (Theorem 9) a constant factor approximation for solving (1).

The greedy Forward Regression (FR) algorithm is the following.
1: $S_0 \leftarrow \emptyset$ and $U \leftarrow \{X_1, \ldots, X_n\}$.
2: In each step $i + 1$, select $X_j \in U \setminus S_i$ maximizing $g(S_i \cup \{X_j\})$. Set $S_{i+1} \leftarrow S_i \cup \{X_j\}$ and $U \leftarrow U \setminus \{X_j\}$.
3: Output $S_k$.

**Theorem 5** *For any set $T$ such that $|T| \leq k$, the set $S$ selected by the greedy FR algorithm satisfies* $g(S) = R_Z^2(S) + f(S) \geq (1 - e^{-\frac{\gamma_{S,2k}}{2}}) g(S \cup T)$.

The proof is very similar to that of [5, Theorem 3.2] and is omitted due to space constraints. Next, we consider the problem of unconstrained maximization of the function $g(S) = R_Z^2(S) + f(S)$. For this, we use a local search (LS) algorithm similar to [9].
1: $S \leftarrow \mathrm{argmax}_i f(X_i)$ and $U \leftarrow \{X_1, \ldots, X_n\}$.
2: If there exists an element $x \in U \setminus S$ such that $f(S \cup \{x\}) \geq (1 + \frac{\epsilon}{n^2}) f(S)$, then set $S \leftarrow S \cup \{x\}$, and go back to Step 2.
3: Output $\mathrm{argmax}_{T \in \{S, U \setminus S, U\}} g(T)$.

Notice that even though we are interested in maximizing $g(S)$, our LS algorithm finds a local optima using $f$, but then uses $g$ to compute the maximum in the last step. To analyze the performance guarantees of LS, we first use the following result of [9, Theorem 3.4].

**Lemma 6** *If $f$ is non-negative and submodular, then for any set $T \subseteq U$ and any $\epsilon > 0$, the LS algorithm takes $O(\frac{1}{\epsilon} n^3 \log n)$ time and outputs solution $S$ such that $(2 + \frac{2\epsilon}{n}) f(S) + f(U \setminus S) \geq f(T)$.*

Using the above, we prove an approximation guarantee for unconstrained maximization of $g(S)$.

**Theorem 7** *The LS algorithm is a $\frac{1}{4+\frac{4\epsilon}{n}}$ approximation for solving $\mathrm{argmax}_S \, g(S)$.*

**Proof.** Suppose the optimal solution is $C^*$ such that $g(C^*) = \mathrm{OPT}$. Consider the set $S$ obtained by the LS algorithm when it terminates. We obtain $g(C^*) = f(C^*) + R^2(C^*) \leq (2 + 2\epsilon/n)f(S) + f(U \setminus S) + R^2(U) \leq (2 + 2\epsilon/n)g(S) + g(U \setminus S) + g(U)$, where the second step follows from Lemma 6 and the monotonicity of $R^2$ and the last step follows from the non-negativity of $f$ and $R^2$. Thus, $\max(g(S), g(U \setminus S), g(U)) \geq \frac{1}{4+\frac{4\epsilon}{n}} g(C^*)$. ∎

We now present the greedy and local search (GLS) algorithm for solving (1) for any submodular, non-monotone, non-negative regularizer.

1: $U \leftarrow \{X_1, \ldots, X_n\}$.
2: $S_1 \leftarrow \mathrm{FR}(U)$, $\quad S_1' \leftarrow \mathrm{LS}(S_1)$, $\quad S_2 \leftarrow \mathrm{FR}(U \setminus S_1)$.
3: Output $\mathrm{argmax}_{S \in \{S_1, S_1', S_2\}} \, g(S)$.

Next, we prove a multiplicative approximation guarantee for the GLS algorithm.

**Lemma 8** *Given sets $C, S_1 \subseteq U$, let $C' = C \setminus S_1$ and $S_2 \subseteq U \setminus S_1$. Then $g(S_1 \cup C) + g(S_2 \cup C') + g(S_1 \cap C) \geq g(C)$.*

**Proof.** Using the submodularity of $f$ and the monotonicity of $R_Z^2(S)$, we obtain $g(S_1 \cup C) + g(S_2 \cup C') = R_Z^2(S_1 \cup C) + R_Z^2(S_2 \cup C') + f(S_1 \cup C) + f(S_2 \cup C') \geq R_Z^2(C) + f(S_1 \cup S_2 \cup C) + f(C')$. Now, $f(C') + f(S_1 \cap C) \geq f(C) + f(\emptyset) \geq f(C)$, or $f(C') \geq f(C) - f(S_1 \cap C)$. Hence, we have $g(S_1 \cup C) + g(S_2 \cup C') + f(S_1 \cap C) \geq R_Z^2(C) + f(C) = g(C)$. ∎

**Theorem 9** *If $f$ is non-negative and submodular and $\epsilon < \frac{n}{4}$, the set $\tilde{S}$ selected by the GLS algorithm gives a $\dfrac{1 - e^{-\frac{\gamma_{\tilde{S},2k}}{2}}}{2 + (1 - e^{-\frac{\gamma_{\tilde{S},2k}}{2}})(4 + 4\epsilon/n)} \geq \dfrac{1 - e^{-\frac{\gamma_{\tilde{S},2k}}{2}}}{7}$ approximation for solving $\mathrm{argmax}_{S:|S| \leq k} \, g(S)$.*

**Proof.** Let $C^*$ be the optimal solution with $g(C^*) = \mathrm{OPT}$. Then $g(S_1) \geq \kappa g(S_1 \cup C^*)$, where $\kappa = (1 - e^{-\frac{\gamma_{S_1,2k}}{2}})$. If $g(S_1 \cap C^*) \geq \epsilon \mathrm{OPT}$, then using the LS algorithm on $S_1$, we get (using Theorem 7) a solution of value at least $\frac{\epsilon}{\alpha} g(C^*)$, where $\alpha = 4 + \frac{4\epsilon}{n}$. Else, $g(S_1) \geq \kappa g(S_1 \cup C^*) + \kappa g(S_1 \cap C^*) - \kappa \epsilon \mathrm{OPT}$. Also, $g(S_2) \geq \kappa g(S_2 \cup (C^* \setminus S_1))$. Thus, $g(S_1) + g(S_2) \geq \kappa g(S_1 \cup C^*) + \kappa g(S_1 \cap C^*) - \kappa \epsilon \mathrm{OPT} + \kappa g(S_2 \cup (C^* \setminus S_1)) \geq \kappa g(C^*) - \kappa \epsilon \mathrm{OPT} \geq \kappa(1 - \epsilon)\mathrm{OPT}$, where the last inequality follows from Lemma 8. Thus, $\max(g(S_1), g(S_2)) \geq \frac{\kappa(1-\epsilon)\mathrm{OPT}}{2}$. Hence, the approximation factor is $\max(\frac{\epsilon}{\alpha}, \frac{\kappa(1-\epsilon)}{2})$. Setting $\epsilon = \frac{\kappa\alpha}{\kappa\alpha + 2}$, we get a $\frac{\kappa}{\kappa\alpha + 2}$-approximation. ∎

When $f(S)$ is a *monotone*, non-negative, submodular function, the problem becomes much easier due to the proposition below that follows directly from the definition of the submodularity ratio.

**Proposition 10** *For any submodular set function $f(S)$, the function $g(S) = R_Z^2(S) + f(S)$ satisfies $\gamma_{U,k}(g) \geq \gamma_{U,k}(R^2)$ for any $U$ and $k$.*

Thus, since $g(S)$ is monotone and approximately submodular, we can directly apply [4, Theorem 3] to show that the greedy FR algorithm gives a $(1 - e^{-\gamma_{\tilde{S},k}(f)})$-approximation.

# 5 Spectral regularizers for diversity

In this section we propose a number of diversity-promoting regularizers for the feature selection problem. We then prove that our algorithms in the previous section can obtain provable guarantees for each of the corresponding regularized feature selection problems.

Most of our analysis requires the notion of operator antitone function [1] and its connection with submodularity that was recently obtained by Friedland and Gaubert [10].

**Definition 11 (Operator antitone functions [1])** *A real valued function $h$ is* operator antitone *on the interval $\Gamma \in R$ if for all $n \geq 1$ and for all $n \times n$ Hermitian matrices $A$ and $B$, we have $A \preceq B \implies h(B) \preceq h(A)$, where $A \preceq B$ denotes that $B - A$ is positive semidefinite; the function $h$ is called* operator monotone *if $-h$ is operator antitone.*

**Theorem 12 ([10])** *Let $f$ be a real continuous function defined on an interval $\Gamma$ of $\mathbb{R}$. If the derivative of $f$ is operator antitone on the interior of $\Gamma$, then for every $n \times n$ Hermitian matrix $C$ with spectrum in $\Gamma$, the set function (from $2^n \longrightarrow \mathbb{R}$) $\operatorname{tr}(f(S)) = \sum_{i=1}^{n} f(\lambda_i(C_S))$ is submodular.*

We will frequently use the following lemma for proving monotonicity of set functions. The proof is given in the supplementary material.

**Lemma 13** *If $f$ is a monotone and non-negative function defined on $\mathbb{R}$, then for every $n \times n$ Hermitian matrix $C$, the set function $\operatorname{tr}(f(S)) = \sum_{i=1}^{n} f(\lambda_i(C_S))$ is monotone.*

## 5.1 Smoothed differential entropy regularizer

For any set $S$ with the corresponding covariance matrix $C_S$, we define the *smoothed differential entropy regularizer* as $f_{\mathrm{de}}(S) = \sum_{i=1}^{|S|} \log_2(\delta + \lambda_i(C_S)) - 3k \log_2 \delta$, where $\delta > 0$ is the *smoothing constant*. This is a smoothed version of the log-determinant function $f_{\mathrm{ld}}(S) = \log(\det(C_S)) = \sum_{i=1}^{|S|} \log(\lambda_i(C_S))$, that is also normalized by an additive term of $3k \log_2 \delta$ in order to make the regularizer non-negative [3].

As shown in Lemma 4, this regularizer also helps improve the robustness of the regression model to noise since maximizing $f_{\mathrm{ld}}(S)$ minimizes the entropy of the perturbation error. For a multivariate Gaussian distribution, $f_{\mathrm{ld}}(S)$ also equivalent (up to an additive $|S|$ factor) to the differential entropy of $S$. However, $f_{\mathrm{ld}}(S)$ is undefined if $S$ is rank-deficient and might also take negative values; the smoothed version $f_{\mathrm{de}}(S)$ overcomes these issues. It is also easy to show that $f_{\mathrm{de}}(S)$ is a diversity-promoting regularizer. We now show that the GLS algorithm to solve (1) with $f(S) = f_{\mathrm{de}}(S)$ gives a constant-factor approximation algorithm.

**Theorem 14** *The set $\tilde{S}$ selected by the GLS algorithm gives a $\frac{1 - e^{-\frac{\gamma_{\tilde{S},2k}}{2}}}{7}$ multiplicative approximation guarantee for (1) using the smoothed differential entropy regularizer $f_{\mathrm{de}}(S)$.*

**Proof.** We first prove that $f_{\mathrm{de}}(S)$ is non-negative and submodular. Consider the real-valued function $\tilde{f}(t) = \log(\delta + t)$ defined on the appropriate interval of $\mathbb{R}$. We will show that the derivative of $\tilde{f}$ is operator antitone. Let $A, B$ be $k \times k$ Hermitian matrices, such that $0 \prec A \preceq B$. Let $I$ denote the identity matrix. Then $A + \delta I \preceq B + \delta I$. Taking inverses, $(B + \delta I)^{-1} \preceq (A + \delta I)^{-1}$. Thus, by Definition 11, the function $h(t) = \frac{1}{\delta + t}$ is operator antitone. Since $h(t)$ is the derivative of $\tilde{f}(t)$, a straightforward application of Theorem 12 gives us that $f_{\mathrm{de}}(S)$ is submodular. By Proposition 10, we obtain that $g(S)$ is approximately submodular, with submodularity ratio at least $\gamma_{\tilde{S},k}(R^2)$. Since $g(S)$ is also non-negative, we can now apply Theorem 9 to obtain the approximation guarantee. ∎

Notice that since $f_{\mathrm{de}}(S)$ is not monotone in general [13], we cannot use Theorem 3. However, in the case when $\delta \geq 1$, a simple application of Lemma 13 shows that $f_{\mathrm{de}}(S)$ becomes monotonically increasing and we can then use Theorem 3 to obtain a tighter approximation bound.

## 5.2 Generalized rank regularizer

For any set $S$ with covariance matrix $C_S$, and constant $\alpha$ such that $0 \leq \alpha \leq 1$, we define the *generalized rank regularizer* as $f_{\mathrm{gr}}(S) = \sum_{i=1}^{|S|} \lambda_i(C_S)^\alpha$. Notice that for $\alpha = 0$, $f_{\mathrm{gr}}(S) = \operatorname{rank}(C_S)$. The rank function however, does not discriminate between a full-rank matrix and an orthogonal matrix, and hence we define $f_{\mathrm{gr}}(S)$ as a generalization of the rank function. It is easy to show that $f_{\mathrm{gr}}(S)$ is diversity-promoting. We prove that $f_{\mathrm{gr}}(S)$ is also monotone and submodular, and hence obtain approximation guarantees for the greedy FR algorithm for (1) with $f(S) = f_{\mathrm{gr}}(S)$.

**Theorem 15** *The set $\tilde{S}$ selected by the greedy FR algorithm gives a $(1 - e^{-\gamma_{\tilde{S},k}(R^2)})$ multiplicative approximation guarantee for (1) using the generalized rank regularizer $f_{\mathrm{gr}}(S)$.*

**Proof.** Consider the real-valued function $\tilde{f}(t) = t^\alpha$ defined on $t \in \mathbb{R}$. It is well known [1] that the derivative of $\tilde{f}$ is operator antitone. Thus, Theorem 12 gives us that $f_{\mathrm{gr}}(S)$ is submodular. Hence, by applying Lemma 10, we obtain that $g(S)$ is an approximately submodular function, with submodularity ratio at least $\gamma_{\tilde{S},k}(R^2)$. Also, by definition $\tilde{f}(t)$ is non-negative and monotone. Thus, using Lemma 13, we get that $f_{\mathrm{gr}}(S)$ and consequently $g(S)$ is a monotonically increasing set function. Since $g(S)$ is non-negative, monotone, and submodular, we can now apply Theorem 3 to obtain a $(1 - e^{-\gamma_{\tilde{S},k}(R^2)})$ approximation ratio. ∎

## 5.3 Spectral variance regularizer

For a set $S$ with covariance matrix $C_S$, we define the *spectral variance regularizer* as $-\sum_{i=1}^{|S|}(\lambda_i(C_S) - 1)^2$. This regularizes the variance of the eigenvalues of the matrix (recall that for an orthogonal matrix, all the eigenvalues are equal to 1) and can be shown to be diversity-promoting. For non-negativity, we add a constant $9k^2$ term[4] to the regularizer and define $f_{\mathrm{sv}}(S) = 9k^2 - \sum_{i=1}^{|S|}(\lambda_i(C_S) - 1)^2$. As with $f_{\mathrm{de}}(S)$, we can show (proof relegated to the supplementary material) that $f_{\mathrm{sv}}(S)$ is submodular, but it is not monotonically increasing in general. Hence, appealing to Theorem 9, we obtain the following.

**Theorem 16** *The set $\tilde{S}$ selected by the GLS algorithm gives a $\frac{1 - e^{-\frac{\gamma_{\tilde{S},2k}}{2}}}{7}$ multiplicative approximation guarantee for (1) using the spectral variance regularizer $f_{\mathrm{sv}}(S)$.*

# 6 Experiments and results

In this section we conduct experiments in different settings to validate the robustness of our spectral regularizers. We compare our approach against two baselines: Lasso and greedy FR. We use two different datasets for the experiments, the `mnist` data (`http://yann.lecun.com/exdb/mnist/`) and a `simulation` data (for which, results are presented in the supplementary material).

The way we synthesize a regression problem out of the `mnist` dataset is as follows. Each image is regarded as a feature vector (of size 784) consisting of the pixel intensities. The target vector for the regression problem consists of the vector of labels. We only sample 1000 images out of the set, and thus have a regression problem with $X \in \mathbb{R}^{1000 \times 784}$ and $Z \in \mathbb{R}^{1000}$. We then preprocess the columns of matrix $X$ and the target vector $Z$ to have unit $\ell_2$-length.

We use two baselines: `lasso` and `no-reg`, the greedy FR with no regularizer. We also use four different spectral regularizers: `ld` (Section 5.1, with $\delta = 1$), `ld-0.1` (Section 5.1, with $\delta = 0.1$), `sv` (Section 5.3), and `gr` (Section 5.2). We considered two different types of perturbations: perturbing $Z$ and $X$. In order to perturb $Z$, we first sample a random vector $\eta \in \mathbb{R}^{1000}$, $\eta_i \sim N(0,1)$. We then create $Z' = Z + \sigma \frac{\eta}{\|\eta\|}$, where $\sigma$ is varied in $[0,1]$[5]. If $S$ is the set of features selected, then the unperturbed regression coefficients are defined as $\alpha = C_S^{-1} X_S^T Z$, and the perturbed coefficients as $\alpha' = C_S^{-1} X_S^T Z'$. The error that we measure is $\|\alpha - \alpha'\|_2$. Similarly, in order to perturb $X$, we first sample $E \in \mathbb{R}^{1000 \times 784}$. Let $E_{\star i}$ denote the $i$th column of $E$. Then, we create $X'$, the perturbed version of $X$ columnwise as $X'_{\star i} = X_{\star i} + \sigma \frac{E_{\star i}}{\|E_{\star i}\|}$. Here again, the perturbed regression coefficients are $\alpha' = C_S'^{-1} X_S'^T y$ where $C_S' = (X_S')^T X_S'$ and the error is measured as $\|\alpha - \alpha'\|_2$. For our experiments, we apply each random perturbation 5 times and then take the average error. Note that the differential entropy of $\alpha - \alpha'$ is directly given by Lemma 4; we will directly measure the quantity on the RHS of the equation of Lemma 4.

**Results.** Figure 1 summarizes the result for the `mnist` data. For clarity of presentation, we have only shown the results of greedy FR for monotone regularizers (`ld` and `gr`) and GLS for non-monotone (`ld-0.1`, `sv`). We also show the results only for $\sigma = 0.1$; results for other values of $\sigma$ are similar. The way we decided on the regularization parameters $\lambda$ is as follows. First we run the `lasso` using a regularization path approach, and obtain a set of solutions for a range of

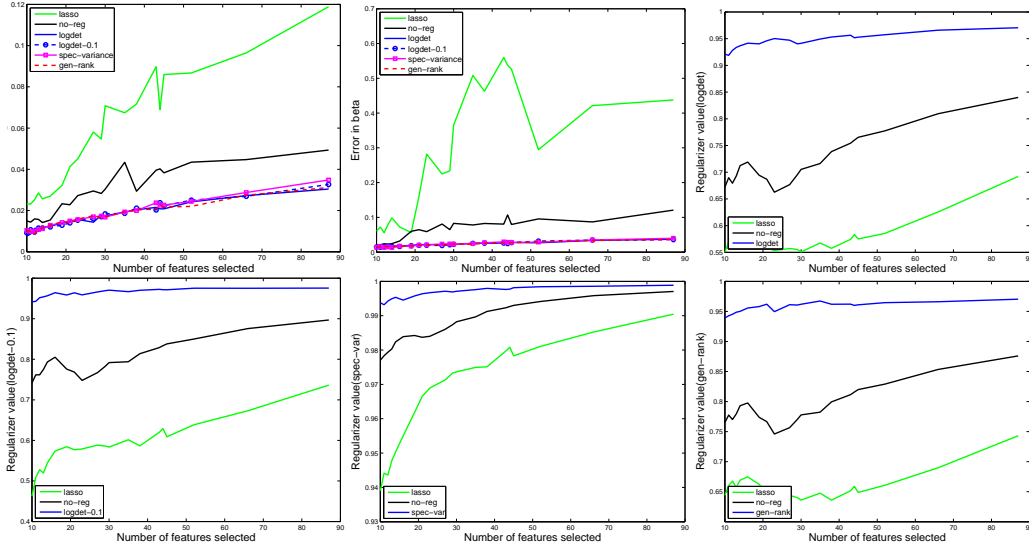

Figure 1: All plots on `mnist` data. (a) Error when $Z$ is perturbed ($\sigma = 0.1$). (b) Error when $X$ is perturbed ($\sigma = 0.1$). (c) Diversity comparison for `ld`. (d) Diversity comparison for `ld-0.1`. (e) Diversity comparison for `sv`. (f) Diversity comparison for `gr`.

.

regularization parameter values and corresponding sparsity ($k$) values. For the other algorithms, we use each of this set of sparsity values as the target number of features to be selected. We chose the regularization constant ($\nu$) to be the maximum subject to the condition that the $R^2$ value for that solution should be greater than that obtained by the `lasso` solution with corresponding sparsity. This ensures we are not sacrificing diversity for solution quality.

Figure 1(a) shows the errors obtained when perturbing the $Z$ vector. As is obvious from the figure, the coefficient vector obtained by `lasso` is very susceptible to perturbation, and the effect of perturbation increases with the number of features used by `lasso`. This indicates that as `lasso` starts incorporating more features, it does not ensure that the features are diverse enough so as to be robust to perturbation. Greedy with no regularization seems more stable than `lasso` but still shows an increasing trend. On the other hand, the errors obtained by perturbing is much less for any of the regularizers, and is only very mildly increasing with $k$: it does not seem to matter which regularizer we employ. Figure 1(b) shows the error obtained when perturbing the $X$ matrix; the same story is true here also. In both cases, using any of the regularizers we are able to pick a set of features that are more robust to perturbation.

Figures 1(c)- 1(f) show that our features are also more diverse than the ones obtained by both `lasso` and `no-reg`. Since there is no one definition of diversity, in each of the plots, we take one of the definitions of diversity value corresponding to the four regularizers we use. In order to be able to compare, the regularizer values for each $k$ are normalized by the maximum value possible for that $k$. For each of the plots we show the values of the diversity value for solutions at different levels of sparsity. It is obvious that we get more diverse solutions than both `lasso` and `no-reg`. The lines corresponding to `lasso` or `no-reg` show an increasing trend because of the normalization.

## 7  Conclusions

In this paper we proposed submodular spectral regularizers for diverse feature selection and obtained efficient approximation algorithms using greedy and local search algorithms. These algorithms obtain a more diverse and noise-insensitive set of features. It would be interesting to see whether we can design convex relaxations for such approaches, and to compare our approach with related ones e.g. CLASH [18] that presents a general framework for merging combinatorial constraints with the L1-norm constraint for LASSO, or with Elastic-Net that provides stability to the features selected when groups of correlated variables are present.

## Footnotes

[1] discussed in the supplementary material at http://cs.usc.edu/~abhimand/nips12supplementary.pdf

[2] We assume throughout that $C_S$ is non-singular. For some of our results, an extension to singular matrices is possible using the Moore–Penrose generalized inverse.

[3] we need this regularizer to be non-negative for sets of size up to $3k$, because of the use of $f(S_1 \cup S_2 \cup C)$ in the proof of Lemma 8

[4]as before, we need this regularizer to be non-negative for sets of size up to $3k$ due to the proof of Lemma 8

[5]Strictly speaking, normalizing $\eta$ makes it non-Gaussian, but for a high dimensional vector $\|\eta\|$ is highly concentrated.

# References

[1] R. Bhatia. *Matrix Analysis*. Springer, 1997.

[2] J. K. Bradley, A. Kyrola, D. Bickson, and C. Guestrin. Parallel coordinate descent for $l1$-regularized loss minimization. In *ICML*, pages 321–328, 2011.

[3] E. J. Candes, J. Romberg, and T. Tao. Stable signal recovery from incomplete and inaccurate measurements. *CPAM*, 59:1207–1223, 2005.

[4] A. Das. *Subset Selection Algorithms for Prediction*. PhD thesis, University of Southern California, 2011.

[5] A. Das and D. Kempe. Submodular meets spectral: Greedy algorithms for subset selection, sparse approximation and dictionary selection. In *ICML*, pages 1057–1064, 2011.

[6] G. Diekhoff. *Statistics for the Social and Behavioral Sciences*. Wm. C. Brown Publishers, 2002.

[7] C. Ding and H. Peng. Minimum redundancy feature selection from microarray gene expression data. In *J. Bioinform. Comput. Biol.*, pages 523–529, 2003.

[8] D. Donoho. For most large underdetermined systems of linear equations, the minimal 11-norm near-solution approximates the sparsest near-solution. *CPAM*, 59:1207–1223, 2005.

[9] U. Feige, V. S. Mirrokni, and J. Vondrak. Maximizing non-monotone submodular functions. *SIAM J. Comput*, 40(4):1133–1153, 2011.

[10] S. Friedland and S. Gaubert. Submodular spectral functions of principal submatrices of a Hermitian matrix, extensions and applications. *Linear Algebra and its Applications*, 2011.

[11] A. Gilbert, S. Muthukrishnan, and M. Strauss. Approximation of functions over redundant dictionaries using coherence. In *SODA*, 2003.

[12] E. Grave, G. Obozinski, and F. R. Bach. Trace Lasso: a trace norm regularization for correlated designs. In *NIPS*, 2011.

[13] C. Guestrin, A. Krause, and A. Singh. Near-optimal sensor placements in Gaussian processes. In *ICML*, 2005.

[14] A. Gupta, A. Roth, G. Schoenebeck, and K. Talwar. Constrained non-monotone submodular maximization: Offline and secretary algorithms. In *WINE*, pages 246–257, 2010.

[15] R. A. Horn and C. R. Johnson. *Matrix Analysis*. Cambridge University Press, 1999.

[16] R. A. Johnson and D. W. Wichern. *Applied Multivariate Statistical Analysis*. Prentice Hall, 2002.

[17] C.-W. Ko, J. Lee, and M. Queyranne. An exact algorithm for maximum entropy sampling. *OR*, 43(4):684–691, 1995.

[18] A. Kyrillidis and V. Cevher. Combinatorial selection and least absolute shrinkage via the clash algorithm. In *Information Theory Proceedings (ISIT), 2012 IEEE International Symposium on*, pages 2216 –2220, july 2012.

[19] A. Miller. *Subset Selection in Regression*. Chapman and Hall, second edition, 2002.

[20] R. Tibshirani. Regression shrinkage and selection via the Lasso. *JRSS*, 58:267–288, 1996.

[21] J. Tropp. Greed is good: Algorithmic results for sparse approximation. *IEEE Trans. Information Theory*, 50:2231–2242, 2004.

[22] J. Tropp. Just relax: Convex programming methods for identifying sparse signals. *IEEE TOIT*, 51:1030–1051, 2006.

[23] L. Yu. Redundancy based feature selection for microarray data. In *SIGKDD*, pages 737–742, 2004.

[24] T. Zhang. Adaptive forward-backward greedy algorithm for sparse learning with linear models. In *NIPS*, 2008.

[25] S. Zhou. Thresholding procedures for high dimensional variable selection and statistical estimation. In *NIPS*, 2009.

